# A Bound on the Error of Cross Validation Using the Approximation and Estimation Rates, with Consequences for the Training-Test Split

**Michael Kearns**
AT&T Research

## 1 INTRODUCTION

We analyze the performance of cross validation [1] in the context of model selection and complexity regularization. We work in a setting in which we must choose the right number of parameters for a hypothesis function in response to a finite training sample, with the goal of minimizing the resulting generalization error. There is a large and interesting literature on cross validation methods, which often emphasizes asymptotic statistical properties, or the exact calculation of the generalization error for simple models. Our approach here is somewhat different, and is primarily inspired by two sources. The first is the work of Barron and Cover [2], who introduced the idea of bounding the error of a model selection method (in their case, the Minimum Description Length Principle) in terms of a quantity known as the *index of resolvability*. The second is the work of Vapnik [5], who provided extremely powerful and general tools for uniformly bounding the deviations between training and generalization errors.

We combine these methods to give a new and general analysis of cross validation performance. In the first and more formal part of the paper, we give a rigorous bound on the error of cross validation in terms of two parameters of the underlying model selection problem: the *approximation rate* and the *estimation rate*. In the second and more experimental part of the paper, we investigate the implications of our bound for choosing $\gamma$, the fraction of data withheld for testing in cross validation. The most interesting aspect of this analysis is the identification of several qualitative properties of the optimal $\gamma$ that appear to be invariant over a wide class of model selection problems:

- When the target function complexity is small compared to the sample size, the performance of cross validation is relatively insensitive to the choice of $\gamma$.

- The importance of choosing $\gamma$ optimally increases, and the optimal value for $\gamma$ decreases, as the target function becomes more complex relative to the sample size.

- There is nevertheless a single *fixed* value for $\gamma$ that works *nearly* optimally for a wide range of target function complexity.

## 2 THE FORMALISM

We consider model selection as a two-part problem: choosing the appropriate number of parameters for the hypothesis function, and tuning these parameters. The training sample is used in both steps of this process. In many settings, the tuning of the parameters is determined by a fixed learning algorithm such as backpropagation, and then model selection reduces to the problem of choosing the architecture. Here we adopt an idealized version of this division of labor. We assume a nested sequence of function classes $H_1 \subset \cdots \subset H_d \cdots$, called the *structure* [5], where $H_d$ is a class of boolean functions of $d$ parameters, each

function being a mapping from some input space $X$ into $\{0, 1\}$. *For simplicity, in this paper we assume that the Vapnik-Chervonenkis (VC) dimension [6, 5] of the class $H_d$ is $O(d)$.* To remove this assumption, one simply replaces all occurrences of $d$ in our bounds by the VC dimension of $H_d$. We assume that we have in our possession a learning algorithm $L$ that on input any training sample $S$ and any value $d$ will output a hypothesis function $h_d \in H_d$ that minimizes the training error over $H_d$ — that is, $\epsilon_t(h_d) = \min_{h \in H_d}\{\epsilon_t(h)\}$, where $\epsilon_t(h)$ is the fraction of the examples in $S$ on which $h$ disagrees with the given label. In many situations, training error minimization is known to be computationally intractable, leading researchers to investigate heuristics such as backpropagation. The extent to which the theory presented here applies to such heuristics will depend in part on the extent to which they approximate training error minimization for the problem under consideration.

Model selection is thus the problem of choosing the best value of $d$. More precisely, we assume an arbitrary *target function* $f$ (which may or may not reside in one of the function classes in the structure $H_1 \subset \cdots \subset H_d \cdots$), and an input distribution $P$; $f$ and $P$ together define the *generalization error* function $\epsilon_g(h) = \text{Pr}_{x \in P}[h(x) \neq f(x)]$. We are given a training sample $S$ of $f$, consisting of $m$ random examples drawn according to $P$ and labeled by $f$ (with the labels possibly corrupted by a noise process that randomly complements each label independently with probability $\eta < 1/2$). The goal is to minimize the *generalization error* of the hypothesis selected.

In this paper, we will make the rather mild but very useful assumption that the structure has the property that for any sample size $m$, there is a value $d_{max}(m)$ such that $\epsilon_t(h_{d_{max}(m)}) = 0$ for any labeled sample $S$ of $m$ examples. We call the function $d_{max}(m)$ the *fitting number* of the structure. The fitting number formalizes the simple notion that with enough parameters, we can always fit the training data perfectly, a property held by most sufficiently powerful function classes (including multilayer neural networks). We typically expect the fitting number to be a linear function of $m$, or at worst a polynomial in $m$. The significance of the fitting number for us is that no reasonable model selection method should choose $h_d$ for $d \geq d_{max}(m)$, since doing so simply adds complexity without reducing the training error.

In this paper we concentrate on the simplest version of cross validation. We choose a parameter $\gamma \in [0, 1]$, which determines the split between training and test data. Given the input sample $S$ of $m$ examples, let $S'$ be the subsample consisting of the first $(1 - \gamma)m$ examples in $S$, and $S''$ the subsample consisting of the last $\gamma m$ examples. In cross validation, rather than giving the entire sample $S$ to $L$, we give only the smaller sample $S'$, resulting in the sequence $h_1, \ldots, h_{d_{max}((1-\gamma)m)}$ of increasingly complex hypotheses. Each hypothesis is now obtained by training on only $(1 - \gamma)m$ examples, which implies that we will only consider values of $d$ smaller than the corresponding fitting number $d_{max}((1 - \gamma)m)$; let us introduce the shorthand $d_{max}^\gamma$ for $d_{max}((1 - \gamma)m)$. Cross validation chooses the $h_d$ satisfying $h_d = \min_{i \in \{1, \ldots, d_{max}^\gamma\}}\{\epsilon_t''(h_i)\}$ where $\epsilon_t''(h_i)$ is the error of $h_i$ on the subsample $S''$. Notice that we are not considering multifold cross validation, or other variants that make more efficient use of the sample, because our analyses will require the independence of the test set. However, we believe that many of the themes that emerge here may apply to these more sophisticated variants as well.

We use $\epsilon_{cv}(m)$ to denote the generalization error $\epsilon_g(h_d)$ of the hypothesis $h_d$ chosen by cross validation when given as input a sample $S$ of $m$ random examples of the target function. Obviously, $\epsilon_{cv}(m)$ depends on $S$, the structure, $f$, $P$, and the noise rate. *When bounding $\epsilon_{cv}(m)$, we will use the expression "with high probability" to mean with probability $1 - \delta$ over the sample $S$, for some small fixed constant $\delta > 0$.* All of our results can also be stated with $\delta$ as a parameter at the cost of a $\log(1/\delta)$ factor in the bounds, or in terms of the expected value of $\epsilon_{cv}(m)$.

## 3  THE APPROXIMATION RATE

It is apparent that any nontrivial bound on $\epsilon_{cv}(m)$ must take account of some measure of the "complexity" of the unknown target function $f$. The correct measure of this complexity is less obvious. Following the example of Barron and Cover's analysis of MDL performance

in the context of density estimation [2], we propose the *approximation rate* as a natural measure of the complexity of $f$ and $P$ in relation to the chosen structure $H_1 \subset \cdots \subset H_d \cdots$. Thus we define the approximation rate function $\epsilon_g(d)$ to be $\epsilon_g(d) = \min_{h \in H_d} \{\epsilon_g(h)\}$. The function $\epsilon_g(d)$ tells us the best generalization error that can be achieved in the class $H_d$, and it is a nonincreasing function of $d$. If $\epsilon_g(s) = 0$ for some sufficiently large $s$, this means that the target function $f$, at least with respect to the input distribution, is realizable in the class $H_s$, and thus $s$ is a coarse measure of how complex $f$ is. More generally, even if $\epsilon_g(d) > 0$ for all $d$, the rate of decay of $\epsilon_g(d)$ still gives a nice indication of how much representational power we gain with respect to $f$ and $P$ by increasing the complexity of our models. Still missing, of course, is some means of determining the extent to which this representational power can be realized by training on a finite sample of a given size, but this will be added shortly. First we give examples of the approximation rate that we will examine following the general bound on $\epsilon_{cv}(m)$.

**The Intervals Problem.** In this problem, the input space $X$ is the real interval $[0, 1]$, and the class $H_d$ of the structure consists of all boolean step functions over $[0, 1]$ of at most $d$ steps; thus, each function partitions the interval $[0, 1]$ into at most $d$ disjoint segments (not necessarily of equal width), and assigns alternating positive and negative labels to these segments. The input space is one-dimensional, but the structure contains arbitrarily complex functions over $[0, 1]$. It is easily verified that our assumption that the VC dimension of $H_d$ is $O(d)$ holds here, and that the fitting number obeys $d_{max}(m) \leq m$. Now suppose that the input density $P$ is uniform, and suppose that the target function $f$ is the function of $s$ alternating segments of equal width $1/s$, for some $s$ (thus, $f$ lies in the class $H_s$). We will refer to these settings as the *intervals problem*. Then the approximation rate is $\epsilon_g(d) = (1/2)(1 - d/s)$ for $1 \leq d < s$ and $\epsilon_g(d) = 0$ for $d \geq s$ (see Figure 1).

**The Perceptron Problem.** In this problem, the input space $X$ is $\Re^N$ for some large natural number $N$. The class $H_d$ consists of all perceptrons over the $N$ inputs in which at most $d$ weights are nonzero. If the input density is spherically symmetric (for instance, the uniform density on the unit ball in $\Re^N$), and the target function is the function in $H_s$ with all $s$ nonzero weights equal to 1, then it can be shown that the approximation rate is $\epsilon_g(d) = (1/\pi)\cos^{-1}(\sqrt{d/s})$ for $d < s$ [4], and of course $\epsilon_g(d) = 0$ for $d \geq s$ (see Figure 1).

**Power Law Decay.** In addition to the specific examples just given, we would also like to study reasonably natural parametric forms of $\epsilon_g(d)$, to determine the sensitivity of our theory to a plausible range of behaviors for the approximation rate. This is important, because in practice we do not expect to have precise knowledge of $\epsilon_g(d)$, since it depends on the target function and input distribution. Following the work of Barron [1], who shows a $c/d$ bound on $\epsilon_g(d)$ for the case of neural networks with one hidden layer under a squared error generalization measure (where $c$ is a measure of target function complexity in terms of a Fourier transform integrability condition) [2], we can consider approximation rates of the form $\epsilon_g(d) = (c/d)^\alpha + \epsilon_{min}$, where $\epsilon_{min} \geq 0$ is a parameter representing the "degree of unrealizability" of $f$ with respect to the structure, and $c, \alpha > 0$ are parameters capturing the rate of decay to $\epsilon_{min}$ (see Figure 1).

## 4   THE ESTIMATION RATE

For a fixed $f$, $P$ and $H_1 \subset \cdots \subset H_d \cdots$, we say that a function $\rho(d, m)$ is an *estimation rate bound* if for all $d$ and $m$, with high probability over the sample $S$ we have $|\epsilon_t(h_d) - \epsilon_g(h_d)| \leq \rho(d, m)$, where as usual $h_d$ is the result of training error minimization on $S$ within $H_d$. Thus $\rho(d, m)$ simply bounds the deviation between the training error and the generalization error of $h_d$. Note that the best such bound may depend in a complicated way on all of the elements of the problem: $f$, $P$ and the structure. Indeed, much of the recent work on the statistical physics theory of learning curves has documented the wide variety of behaviors that such deviations may assume [4, 3]. However, for many natural problems

it is both convenient and accurate to rely on a *universal* estimation rate bound provided by the powerful theory of uniform convergence: Namely, for any $f$, $P$ and any structure, the function $\rho(d,m) = \sqrt{(d/m)\log(m/d)}$ is an estimation rate bound [5]. Depending upon the details of the problem, it is sometimes appropriate to omit the $\log(m/d)$ factor, and often appropriate to refine the $\sqrt{d/m}$ behavior to a function that interpolates smoothly between $d/m$ behavior for small $\epsilon_t$ to $\sqrt{d/m}$ for large $\epsilon_t$. Although such refinements are both interesting and important, many of the qualitative claims and predictions we will make are invariant to them as long as the deviation $|\epsilon_t(h_d) - \epsilon_g(h_d)|$ is well-approximated by a power law $(d/m)^\alpha$ $(\alpha > 0)$; it will be more important to recognize and model the cases in which power law behavior is grossly violated.

Note that this universal estimation rate bound holds only under the assumption that the training sample is noise-free, but straightforward generalizations exist. For instance, if the training data is corrupted by random label noise at rate $0 \le \eta < 1/2$, then $\rho(d,m) = \sqrt{(d/(1-2\eta)^2 m)\log(m/d)}$ is again a universal estimation rate bound.

## 5 THE BOUND

**Theorem 1** *Let $H_1 \subset \cdots \subset H_d \cdots$ be any structure, where the VC dimension of $H_d$ is $O(d)$. Let $f$ and $P$ be any target function and input distribution, let $\epsilon_g(d)$ be the approximation rate function for the structure with respect to $f$ and $P$, and let $\rho(d,m)$ be an estimation rate bound for the structure with respect to $f$ and $P$. Then for any $m$, with high probability*

$$\epsilon_{cv}(m) \le \min_{1\le d\le d_{max}^\gamma} \{\epsilon_g(d) + \rho(d,(1-\gamma)m)\} + O\left(\sqrt{\frac{\log(d_{max}^\gamma)}{\gamma m}}\right) \tag{1}$$

*where $\gamma$ is the fraction of the training sample used for testing, and $d_{max}^\gamma$ is the fitting number $d_{max}((1-\gamma)m)$. Using the universal estimation bound rate and the rather weak assumption that $d_{max}(m)$ is polynomial in $m$, we obtain that with high probability*

$$\epsilon_{cv}(m) \le \min_{1\le d\le d_{max}^\gamma} \left\{\epsilon_g(d) + O\left(\sqrt{\frac{d}{(1-\gamma)m}\log\left(\frac{m}{d}\right)}\right)\right\} + O\left(\sqrt{\frac{\log((1-\gamma)m)}{\gamma m}}\right). \tag{2}$$

*Straightforward generalizations of these bounds for the case where the data is corrupted by classification noise can be obtained, using the modified estimation rate bound given in Section 4* [3].

We delay the proof of this theorem to the full paper due to space considerations. However, the central idea is to appeal twice to uniform convergence arguments: once within each class $H_d$ to bound the generalization error of the resulting training error minimizer $h_d \in H_d$, and a second time to bound the generalization error of the $h_d$ minimizing the error on the test set of $\gamma m$ examples.

In the bounds given by (1) and (2), the $\min\{\cdot\}$ expression is analogous to Barron and Cover's index of resolvability [2]; the final term in the bounds represents the error introduced by the testing phase of cross validation. These bounds exhibit tradeoff behavior with respect to the parameter $\gamma$: as we let $\gamma$ approach 0, we are devoting more of the sample to training the $h_d$, and the estimation rate bound term $\rho(d,(1-\gamma)m)$ is decreasing. However, the test error term $O(\sqrt{\log(d_{max}^\gamma)/(\gamma m)})$ is increasing, since we have less data to accurately estimate the $\epsilon_g(h_d)$. The reverse phenomenon occurs as we let $\gamma$ approach 1.

While we believe Theorem 1 to be enlightening and potentially useful in its own right, we would now like to take its interpretation a step further. More precisely, suppose we

assume that the bound is an approximation to the actual behavior of $\epsilon_{cv}(m)$. Then in principle we can optimize the bound to obtain the best value for $\gamma$. Of course, in addition to the assumptions involved (the main one being that $\rho(d, m)$ is a good approximation to the training-generalization error deviations of the $h_d$), this analysis can only be carried out given information that we should not expect to have in practice (at least in exact form) — in particular, the approximation rate function $\epsilon_g(d)$, which depends on $f$ and $P$. *However, we argue in the coming sections that several interesting qualitative phenomena regarding the choice of $\gamma$ are largely invariant to a wide range of natural behaviors for $\epsilon_g(d)$.*

## 6   A CASE STUDY: THE INTERVALS PROBLEM

We begin by performing the suggested optimization of $\gamma$ for the intervals problem. Recall that the approximation rate here is $\epsilon_g(d) = (1/2)(1 - d/s)$ for $d < s$ and $\epsilon_g(d) = 0$ for $d \geq s$, where $s$ is the complexity of the target function. Here we analyze the behavior obtained by assuming that the estimation rate $\rho(d, m)$ actually behaves as $\rho(d, m) = \sqrt{d/(1 - \gamma)m}$ (so we are omitting the log factor from the universal bound), and to simplify the formal analysis a bit (but without changing the qualitative behavior) we replace the term $\sqrt{\log((1 - \gamma)m)/(\gamma m)}$ by the weaker $\sqrt{\log(m)/m}$. Thus, if we define the function $F(d, m, \gamma) = \epsilon_g(d) + \sqrt{d/(1 - \gamma)m} + \sqrt{\log(m)/(\gamma m)}$ then following Equation (1), we are approximating $\epsilon_{cv}(m)$ by $\epsilon_{cv}(m) \approx \min_{1 \leq d \leq d^{\gamma}_{max}} \{F(d, m, \gamma)\}$ [4].

The first step of the analysis is to fix a value for $\gamma$ and differentiate $F(d, m, \gamma)$ with respect to $d$ to discover the minimizing value of $d$; the second step is to differentiate with respect to $\gamma$. It can be shown (details omitted) that the optimal choice of $\gamma$ under the assumptions is $\gamma_{opt} = (\log(m)/s)^{1/3}/(1 + (\log(m)/s)^{1/3})$. It is important to remember at this point that despite the fact that we have derived a precise expression for $\gamma_{opt}$, due to the assumptions and approximations we have made in the various constants, any quantitative interpretation of this expression is meaningless. However, we can reasonably expect that this expression captures the qualitative way in which the optimal $\gamma$ changes as the amount of data $m$ changes in relation to the target function complexity $s$. On this score the situation initially appears rather bleak, as the function $(\log(m)/s)^{1/3}/(1 + (\log(m)/s)^{1/3})$ is quite sensitive to the ratio $\log(m)/s$, which is something we do not expect to have the luxury of knowing in practice. However, it is both fortunate and interesting that $\gamma_{opt}$ does not tell the entire story. In Figure 2, we plot the function $F(s, m, \gamma)$ as a function of $\gamma$ for $m = 10000$ and for several different values of $s$ (note that for consistency with the later experimental plots, the $x$ axis of the plot is actually the training fraction $1 - \gamma$). Here we can observe four important qualitative phenomena, which we list in order of increasing subtlety: (A) When $s$ is small compared to $m$, the predicted error is relatively insensitive to the choice of $\gamma$: as a function of $\gamma$, $F(s, m, \gamma)$ has a wide, flat bowl, indicating a wide range of $\gamma$ yielding essentially the same near-optimal error. (B) As $s$ becomes larger in comparison to the fixed sample size $m$, the relative superiority of $\gamma_{opt}$ over other values for $\gamma$ becomes more pronounced. In particular, large values for $\gamma$ become progressively worse as $s$ increases. For example, the plots indicate that for $s = 10$ (again, $m = 10000$), even though $\gamma_{opt} = 0.524\cdots$ the choice $\gamma = 0.75$ will result in error quite near that achieved using $\gamma_{opt}$. However, for $s = 500$, $\gamma = 0.75$ is predicted to yield greatly suboptimal error. Note that for very large $s$, the bound predicts vacuously large error for all values of $\gamma$, so that the choice of $\gamma$ again becomes irrelevant. (C) Because of the insensitivity to $\gamma$ for $s$ small compared to $m$, there is a *fixed* value of $\gamma$ which seems to yield reasonably good performance for a wide range of values for $s$. This value is essentially the value of $\gamma_{opt}$ for the case where $s$ is large but nontrivial generalization is still possible, since choosing the best value for $\gamma$ is more important there than for the small $s$ case. (D) The value of $\gamma_{opt}$ is decreasing as $s$ increases. This is slightly difficult to confirm from the plot, but can be seen clearly from the precise expression for $\gamma_{opt}$.

In Figure 3, we plot the results of experiments in which labeled random samples of size $m = 5000$ were generated for a target function of $s$ equal width intervals, for $s = 10, 100$ and 500. The samples were corrupted by random label noise at rate $\eta = 0.3$. For each value of $\gamma$ and each value of $d$, $(1 - \gamma)m$ of the sample was given to a program performing training error minimization within $H_d$; the remaining $\gamma m$ examples were used to select the best $h_d$ according to cross validation. The plots show the true generalization error of the $h_d$ selected by cross validation as a function of $\gamma$ (the generalization error can be computed exactly for this problem). Each point in the plots represents an average over 10 trials.

While there are obvious and significant quantitative differences between these experimental plots and the theoretical predictions of Figure 2, the properties (A), (B) and (C) are rather clearly borne out by the data: (A) In Figure 3, when $s$ is small compared to $m$, there is a wide range of acceptable $\gamma$; it appears that any choice of $\gamma$ between 0.10 and 0.50 yields nearly optimal generalization error. (B) By the time $s = 100$, the sensitivity to $\gamma$ is considerably more pronounced. For example, the choice $\gamma = 0.50$ now results in clearly suboptimal performance, and it is more important to have $\gamma$ close to 0.10. (C) Despite these complexities, there does indeed appear to be single value of $\gamma$ — approximately 0.10 — that performs nearly optimally for the entire range of $s$ examined.

The property (D) — namely, that the optimal $\gamma$ decreases as the target function complexity is increased relative to a fixed $m$ — is certainly not refuted by the experimental results, but any such effect is simply too small to be verified. It would be interesting to verify this prediction experimentally, perhaps on a different problem where the predicted effect is more pronounced.

## 7 CONCLUSIONS

For the cases where the approximation rate $\epsilon_g(d)$ obeys either power law decay or is that derived for the perceptron problem discussed in Section 3, the behavior of $\epsilon_{cv}(m)$ as a function of $\gamma$ predicted by our theory is largely the same (for example, see Figure 4). In the full paper, we describe some more realistic experiments in which cross validation is used to determine the number of backpropogation training epochs. Figures similar to Figures 2 through 4 are obtained, again in rough accordance with the theory.

In summary, our theory predicts that although significant quantitative differences in the behavior of cross validation may arise for different model selection problems, the properties (A), (B), (C) and (D) should be present in a wide range of problems. At the very least, the behavior of our *bounds* exhibits these properties for a wide range of problems. It would be interesting to try to identify natural problems for which one or more of these properties is strongly violated; a potential source for such problems may be those for which the underlying learning curve deviates from classical power law behavior [4, 3].

Acknowledgements: I give warm thanks to Yishay Mansour, Andrew Ng and Dana Ron for many enlightening conversations on cross validation and model selection. Additional thanks to Andrew Ng for his help in conducting the experiments.

## Footnotes

[1] Perhaps in conflict with accepted usage in statistics, here we use the term "cross validation" to mean the simple method of saving out an independent test set to perform model selection. Precise definitions will be stated shortly.

[2]Since the bounds we will give have straightforward generalizations to real-valued function learning under squared error, examining behavior for $\epsilon_g(d)$ in this setting seems reasonable.

[3]The main effect of classification noise at rate $\eta$ is the replacement of occurrences in the bound of the sample size $m$ by the smaller "effective" sample size $(1-\eta)^2 m$.

[4]Although there are hidden constants in the $O(\cdot)$ notation of the bounds, it is the *relative* weights of the estimation and test error terms that is important, and choosing both constants equal to 1 is a reasonable choice (since both terms have the same Chernoff bound origins).

## References

[1] A. Barron. Universal approximation bounds for superpositions of a sigmoidal function. *IEEE Transactions on Information Theory*, 19:930–944, 1991.

[2] A. R. Barron and T. M. Cover. Minimum complexity density estimation. *IEEE Transactions on Information Theory*, 37:1034–1054, 1991.

[3] D. Haussler, M. Kearns, H.S. Seung, and N. Tishby. Rigourous learning curve bounds from statistical mechanics. In *Proceedings of the Seventh Annual ACM Confernce on Computational Learning Theory*, pages 76–87, 1994.

[4] H. S. Seung, H. Sompolinsky, and N. Tishby. Statistical mechanics of learning from examples. *Physical Review*, A45:6056–6091, 1992.

[5] V. N. Vapnik. *Estimation of Dependences Based on Empirical Data*. Springer-Verlag, New York, 1982.

[6] V. N. Vapnik and A. Y. Chervonenkis. On the uniform convergence of relative frequencies of events to their probabilities. *Theory of Probability and its Applications*, 16(2):264–280, 1971.

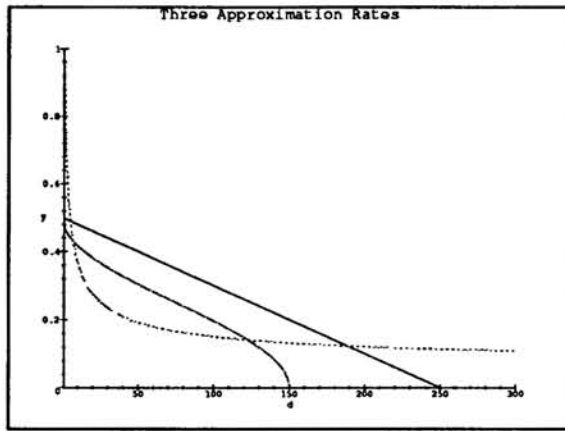

Figure 1: Plots of three approximation rates: for the intervals problem with target complexity $s = 250$ intervals (linear plot intersecting $d$-axis at 250), for the perceptron problem with target complexity $s = 150$ nonzero weights (nonlinear plot intersecting $d$-axis at 150), and for power law decay asymptoting at $\epsilon_{min} = 0.05$.

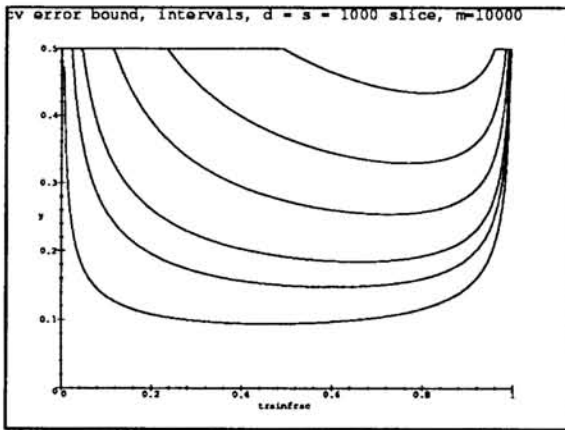

Figure 2: Plot of the predicted generalization error of cross validation for the intervals model selection problem, as a function of the fraction $1 - \gamma$ of data used for training. (In the plot, the fraction of training data is 0 on the left ($\gamma = 1$) and 1 on the right ($\gamma = 0$)). The fixed sample size $m = 10,000$ was used, and the 6 plots show the error predicted by the theory for target function complexity values $s = 10$ (bottom plot), 50, 100, 250, 500, and 1000 (top plot).

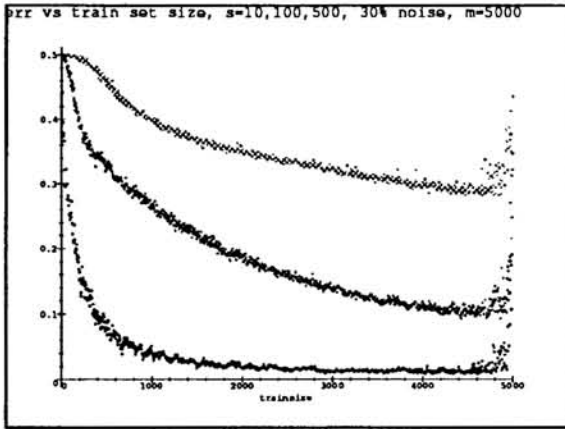

Figure 3: Experimental plots of cross validation generalization error in the intervals problem as a function of training set size $(1-\gamma)m$. Experiments with the three target complexity values $s = 10, 100$ and 500 (bottom plot to top plot) are shown. Each point represents performance averaged over 10 trials.

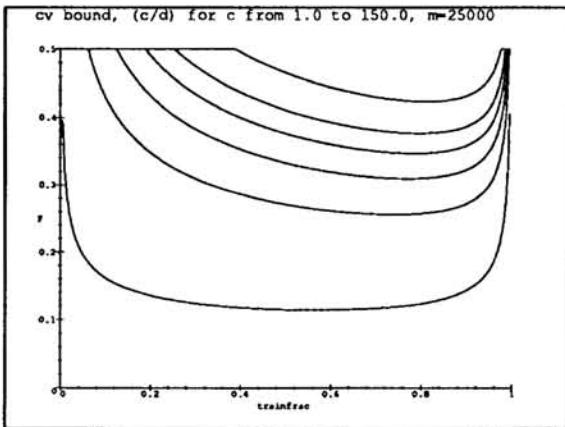

Figure 4: Plot of the predicted generalization error of cross validation for the power law case $\epsilon_g(d) = (c/d)$, as a function of the fraction $1-\gamma$ of data used for training. The fixed sample size $m = 25,000$ was used, and the 6 plots show the error predicted by the theory for target function complexity values $c = 1$ (bottom plot), 25, 50, 75, 100, and 150 (top plot).